# Neural Network Implementation Approaches
## for the
## Connection Machine

Nathan H. Brown, Jr.

MRJ/Perkin Elmer, 10467 White Granite Dr. (Suite 304), Oakton, Va. 22124

## ABSTRACT

The SIMD parallelism of the Connection Machine (CM) allows the construction of neural network simulations by the use of simple data and control structures. Two approaches are described which allow parallel computation of a model's nonlinear functions, parallel modification of a model's weights, and parallel propagation of a model's activation and error. Each approach also allows a model's interconnect structure to be physically dynamic. A Hopfield model is implemented with each approach at six sizes over the same number of CM processors to provide a performance comparison.

## INTRODUCTION

Simulations of neural network models on digital computers perform various computations by applying linear or nonlinear functions, defined in a program, to weighted sums of integer or real numbers retrieved and stored by array reference. The numerical values are model dependent parameters like time averaged spiking frequency (activation), synaptic efficacy (weight), the error in error back propagation models, and computational temperature in thermodynamic models. The interconnect structure of a particular model is implied by indexing relationships between arrays defined in a program. On the Connection Machine (CM), these relationships are expressed in hardware processors interconnected by a 16-dimensional hypercube communication network. Mappings are constructed to define higher dimensional interconnectivity between processors on top of the fundamental geometry of the communication network. Parallel transfers are defined over these mappings. These mappings may be dynamic. CM parallel operations transform array indexing from a temporal succession of references to memory to a single temporal reference to spatially distributed processors.

Two alternative approaches to implementing neural network simulations on the CM are described. Both approaches use "data parallelism"[1] provided by the *Lisp virtual machine. Data and control structures associated with each approach and performance data for a Hopfield model implemented with each approach are presented.

## DATA STRUCTURES

The functional components of a neural network model implemented in *Lisp are stored in a uniform parallel variable (pvar) data structure on the CM. The data structure may be viewed as columns of pvars. Columns are given to all CM virtual processors. Each CM physical processor may support 16 virtual processors. In the first approach described, CM processors are used to represent the edge set of a models graph structure. In the second approach described, each processor can represent a unit, an outgoing link, or an incoming link in a model's structure. Movement of activation (or error) through a model's interconnect structure is simulated by moving numeric values over the CM's hypercube. Many such movements can result from the execution of a single CM macroinstruction. The CM transparently handles message buffering and collision resolution. However, some care is required on the part of the user to insure that message traffic is distributed over enough processors so that messages don't stack up at certain processors, forcing the CM to sequentially handle large numbers of buffered messages. Each approach requires serial transfers of model parameters and states over the communication channel between the host and the CM at certain times in a simulation.

The first approach, "the edge list approach," distributes the edge list of a network graph to the CM, one edge per CM processor. Interconnect weights for each edge are stored in the memory of the processors. An array on the host machine stores the current activation for all units. This approach may be considered to represent abstract synapses on the CM. The interconnect structure of a model is described by product sets on an ordered pair of identification (id) numbers, rid and sid. The rid is the id of units receiving activation and sid the id of units sending activation. Each id is a unique integer. In a hierarchical network, the ids of input units are never in the set of rids and the ids of output units are never in the set of sids. Various set relations (e.g. inverse, reflexive, symmetric, etc.) defined over id ranges can be used as a high level representation of a network's interconnect structure. These relations can be translated into pvar columns. The limits to the interconnect complexity of a simulated model are the virtual processor memory limits of the CM configuration used and the stack space required by functions used to compute the weighted sums of activation. Fig. 1 shows a $R^3 \to R^2 \to R^4$ interconnect structure and its edge list representation on the CM.

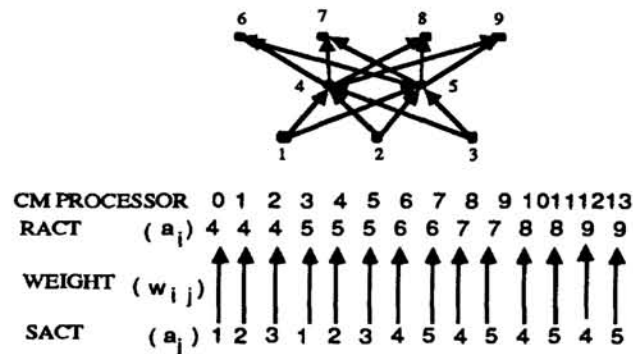

Fig. 1. Edge List Representation of a $R^3 \to R^2 \to R^4$ Interconnect Structure

This representation can use as few as six pvars for a model with Hebbian adaptation: rid (i), sid (j), interconnect weight ($w_{ij}$), ract ($a_i$), sact ($a_j$), and learn rate ($\eta$). Error back propagation requires the addition of: error ($e_i$), old interconnect weight ($w_{ij}(t-1)$), and the momentum term ($\alpha$). The receiver and sender unit identification pvars are described above. The interconnect weight pvar stores the weight for the interconnect. The activation pvar, sact, stores the current activation, $a_j$, transfered to the unit specified by rid from the unit specified by sid. The activation pvar, ract, stores the current weighted activation $a_j w_{ij}$. The error pvar stores the error for the unit specified by the sid. A variety of proclaims (e.g. integer, floating point, boolean, and field) exist in *Lisp to define the type and size of pvars. Proclaims conserve memory and speed up execution. Using a small number of pvars limits the

amount of memory used in each CM processor so that maximum virtualization of the hardware processors can be realized. Any neural model can be specified in this fashion. Sigma-pi models require multiple input activation pvars be specified. Some edges may have a different number of input activation pvars than others. To maintain the uniform data structure of this approach a tag pvar has to be used to determine which input activation pvars are in use on a particular edge.

The edge list approach allows the structure of a simulated model to "physically" change because edges may be added (up to the virtual processor limit), or deleted at any time without affecting the operation of the control structure. Edges may also be placed in any processor because the subselection (on rid or sid) operation performed before a particular update operation insures that all processors (edges) with the desired units are selected for the update.

The second simulation approach, "the composite approach," uses a more complicated data structure where units, incoming links, and outgoing links are represented. Update routines for this approach use parallel segmented scans to form the weighted sum of input activation. Parallel segmented scans allow a MIMD like computation of the weighted sums for many units at once. Pvar columns have unique values for unit, incoming link, and outgoing link representations. The data structures for input units, hidden units, and output units are composed of sets of the three pvar column types. Fig. 2 shows the representation for the same model as in Fig. 1 implemented with the composite approach.

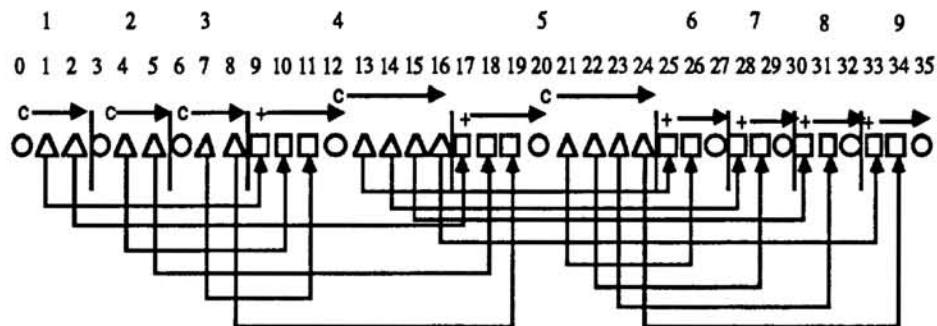

Fig. 2. Composite Representation of a $R^3$ -> $R^2$ -> $R^4$ Interconnect Structure

In Fig. 2, CM processors acting as units, outgoing links, and incoming links are represented respectively by circles, triangles, and squares. CM cube address pointers used to direct the parallel transfer of activation are shown by arrows below the structure. These pointers define the model interconnect mapping. Multiple sets of these pointers may be stored in seperate pvars. Segmented scans are represented by operation-arrow icons above the structure. A basic composite approach pvar set for a model with Hebbian adaptation is: forward B, forward A, forward transfer address, interconnect weight ($w_{ij}$), act-1 ($a_i$), act-2 ($a_j$), threshold, learn rate ($\eta$), current unit id (i), attached unit id (j), level, and column type. Back progagation of error requires the addition of: backward B, backward A, backward transfer address, error ($e_i$), previous interconnect weight ($w_{ij}(t-1)$), and the momentum term ($\alpha$). The forward and backward boolean pvars control the segmented scanning operations over unit constructs. Pvar A of each type controls the plus scanning and pvar B of each type controls the copy scanning. The forward transfer pvar stores cube addresses for

forward (ascending cube address) parallel transfer of activation. The backward transfer pvar stores cube addresses for backward (descending cube address) parallel transfer of error. The interconnect weight, activation, and error pvars have the same functions as in the edge list approach. The current unit id stores the current unit's id number. The attached unit id stores the id number of an attached unit. This is the edge list of the network's graph. The contents of these pvars only have meaning in link pvar columns. The level pvar stores the level of a unit in a hierarchical network. The type pvar stores a unique arbitrary tag for the pvar column type. These last three pvars are used to subselect processor ranges to reduce the number of processors involved in an operation.

Again, edges and units can be added or deleted. Processor memories for deleted units are zeroed out. A new structure can be placed in any unused processors. The level, column type, current unit id, and attached unit id values must be consistent with the desired model interconnectivity.

The number of CM virtual processors required to represent a given model on the CM differs for each approach. Given N units and $N(N-1)$ non-zero interconnects (e.g. a symmetric model), the edge list approach simply distributes $N(N-1)$ edges to $N(N-1)$ CM virtual processors. The composite approach requires two virtual processors for each interconnect and one virtual processor for each unit or $N+2\ N(N-1)$ CM virtual processors total. The difference between the number of processors required by the two approaches is $N^2$. Table I shows the processor and CM virtualization requirements for each approach over a range of model sizes.

TABLE I Model Sizes and CM Processors Required

| Run No. | Grid Size | Number of Units | Edge List $N(N-1)$ | Quart CM Virt. Procs. | Virt. LeveL |
|---|---|---|---|---|---|
| 1 | $8^2$ | 64 | 4032 | 8192 | 0 |
| 2 | $9^2$ | 81 | 6480 | 8192 | 0 |
| 3 | $11^2$ | 121 | 14520 | 16384 | 0 |
| 4 | $13^2$ | 169 | 28392 | 32768 | 2 |
| 5 | $16^2$ | 256 | 65280 | 65536 | 4 |
| 6 | $19^2$ | 361 | 129960 | 131072 | 8 |

| Run No. | Grid Size | Number of Units | Composite $N+2N(N-1)$ | Quart CM Virt. Procs. | Virt. LeveL |
|---|---|---|---|---|---|
| 7 | $8^2$ | 64 | 8128 | 8192 | 0 |
| 8 | $9^2$ | 81 | 13041 | 16384 | 0 |
| 9 | $11^2$ | 121 | 29161 | 32768 | 2 |
| 10 | $13^2$ | 169 | 56953 | 65536 | 4 |
| 11 | $16^2$ | 256 | 130816 | 131072 | 8 |
| 12 | $19^2$ | 361 | 260281 | 262144 | 16 |

## CONTROL STRUCTURES

The control code for neural network simulations (in *Lisp or C*) is stored and executed sequentially on a host computer (e.g. Symbolics 36xx and VAX 86xx) connected to the CM by a high speed communication line. Neural network simulations executed in *Lisp use a small subset of the total instruction set: processor selection reset (*all), processor selection (*when), parallel content assignment (*set), global summation (*sum), parallel multiplication (*!! ), parallel summation (+!!), parallel exponentiation (exp!!), the parallel global memory references (*pset) and (pref!!), and the parallel segmented scans (copy!! and +!!). Selecting CM processors puts them in a "list of active processors" (loop) where their contents may be arithmetically manipulated in parallel. Copies of the list of active processors may be made and used at any time. A subset of the processors in the loop may be "subselected" at any time, reducing the loop contents. The processor selection reset clears the current selected set by setting all processors as selected. Parallel content assignment allows pvars in the currently selected processor set to be assinged allowed values in one step. Global summation executes a tree reduction sum across the CM processors by grid or cube address for particular pvars. Parallel multiplications and additions multiply and add pvars for all selected CM processors in one step. The parallel exponential applies the function, $e^x$, to the contents of a specified pvar, x, over all selected processors. Parallel segmented scans apply two functions, copy!! and +!!, to subsets of CM processors by scanning across grid or cube addresses. Scanning may be forward or backward (i.e. by ascending or descending cube address order, respectively).

Figs. 3 and 4 show the edge list approach kernels required for Hebbian learning for a $R^2 \rightarrow R^2$ model. The loop construct in Fig. 3 drives the activation update

$$a_i(t+1) = F[\Sigma w_{ij}(t+1)a_j(t)] \qquad (1)$$

operation. The usual loop to compute each weighted sum for a particular unit has been replaced by four parallel operations: a selection reset (*all), a subselection of all the processors for which the particular unit is a receiver of activation (*when (=!! rid (!! (1+ u)))), a parallel multiplication (*!! weight sact), and a tree reduction sum (*sum ...). Activation is spread for a particular unit, to all others it is connected to, by: storing the newly computed activation in an array on the host, then subselecting the processors where the particular unit is a sender of activation (*when (=!! sid (!! (1+ u)))), and broadcasting the array value on the host to those processors.

```
(dotimes (u 4)
    (*all (*when (=!! rid (!! (1+ u)))
        (setf (aref activation u)
                (some-nonlinearity (*sum (*!! weight sact))))
        (*set ract (!! (aref activation u)))
    (*all (*when (=!! sid (!! (1+ u)))
        (*set sact (!! (aref activation u)))))))
```

Fig. 3. Activation Update Kernel for the Edge Lst Approach.

Fig. 4 shows the Hebbian weight update kernel

$$w_{ij}(t+1)=\eta a_i(t+1)a_j(t+1). \qquad (2)$$

```
(*all
    (*set weight
        (*!! learn-rate ract sact))))
```

Fig. 4. Hebbian Weight Modification Kernel for the Edge List Approach

The edge list activation update kernel is essentially serial because the steps involved can only be applied to one unit at a time. The weight modification is parallel. For error back propagation a seperate loop for computing the errors for the units on each layer of a model is required. Activation update and error back propagation also require transfers to and from arrays on the host on every iteration step incurring a concomitant overhead.

Other common computations used for neural networks can be computed in parallel using the edge list approach. Fig. 5 shows the code kernel for parallel computation of Lyapunov engergy equations

$$E= -1/2\Sigma^N_{i\neq j}w_{ij}a_ia_j + \Sigma^N_{i=1}I_ia_i \qquad (3)$$

where i=1 to number of units (N).

```
(+ (* -.5 (*sum (*!! weight ract sact))) (*sum (*!! input sact)))
```

Fig. 5. Kernel for Computation of the Lyapunov Energy Equation

Although an input pvar, input, is defined for all edges, it is only non-zero for those edges associated with input units. Fig. 6 shows the pvar structure for parallel computation of a Hopfield weight prescription, with segmented scanning to produce the weights in one step,

$$w_{ij} = \Sigma^S_{r=1}(2a^r_i-1)(2a^r_j-1) \qquad (4)$$

where $w_{ii}=0$, $w_{ij}=w_{ji}$, and r=1 to the number of patterns, S, to be stored.

```
seg
ract     v1_1 v2_1 ... vS_1   v1_1 v2_1 ... vS_1 ...
sact     v1_2 v2_2 ... vS_2   v1_3 v2_3 ... vS_3 ...
weight            w12                  w13
```

Fig. 6. Pvar Structure for Parallel Computation of Hopfield Weight Prescription

Fig. 7 shows the *Lisp kernel used on the pvar structure in Fig. 6.

```
(set weight
    (scan '+!! (*!! (-!! (*!! ract (!! 2)) (!! 1)) (-!! (*!! sact (!! 2)) (!! 1))))
        :segment-pvar seg :include-self t)
```

Fig. 7. Parallel Computation of Hopfield Weight Prescription

The inefficiencies of the edge list activation update are solved by the updating method used in the composite approach. Fig. 8 shows the *Lisp kernel for activation update using the composite approach. Fig. 9 shows the *Lisp kernel for the Hebbian learning operation in the composite approach.

```
(*all
    (*when (=!! level (!! 1))
        (*set act (scan!! act-1 'copy!! :segment-pvar forwardb :include-self t))
        (*set act (*!! act-1 weight))
        (*when (=!! type (!! 2)) (*pset :overwrite act-1 act-1 ftransfer)))
    (*when (=!! level (!! 2))
        (*set act (scan!! act-1 '+!! :segment-pvar forwarda :include-self t))
        (*when (=!! type (!! 1)) (some-nonlinearity!! act-1))))
```

Fig. 8. Activation Update Kernel for the Composite Approach

```
(*all
    (*set act-1 (scan!! act-1 'copy!! :segment-pvar forwardb
                                          :include-self t))
    (*when (=!! type (!! 2))
        (*set act-2 (pref!! act-1 btransfer)))
    (*set weight
            (+!! weight
                (*!! learn-rate act-1 act-2)))))
```

Fig. 9. Hebbian Weight Update Kernel for the Composite Approach

It is immediately obvious that no looping is invloved. Any number of interconnects may be updated by the proper subselection. However, the more subselection is used the less efficient the computation becomes because less processors are invloved.

## COMPLEXITY ANALYSIS

The performance results presented in the next section can be largely anticipated from an analysis of the space and time requirements of the CM implementation approaches. For simplicity I use a $R^n$ -> $R^n$ model with Hebbian adaptation. The oder of magnitude requirements for activation and weight updating are compared for both CM implementation approaches and a basic serial matrix arithmetic approach.

For the given model, the space requirements on a conventional serial machine are $2n+n^2$ locations or $O(n^2)$. The growth of the space requirement is dominated by the nxn weight matrix defining the system interconnect structure. The edge list approach uses six pvars for each processor and uses nxn processors for the mapping, or $6n^2$ locations or $O(n^2)$. The composite approach uses 11 pvars. There are 2n processors for units and $2n^2$ processors for interconnects in the given model. The composite approach uses $11(2n+2n^2)$ locations or $O(n^2)$. The CM implementations take up roughly the same space as the serial implementation, but the space for the serial implementation is composed of passive memory whereas the space for the CM implementations is composed of interconnected processors with memory.

The time analysis for the approaches compares the time order of magnitudes to compute the activation update (1) and the Hebbian weight update (2). On a serial

machine, the n weighted sums computed for the activation update require $n^2$ multiplications and n(n-1) additions. There are $2n^2$-n operations or time order of magnitude $O(n^2)$. The time order of magnitude for the weight matrix update is $O(n^2)$ since there are $n^2$ weight matrix elements.

The edge list approach forms n weighted sums by performing a parallel product of all of the weights and activations in the model, (*!! weight sact), and then a tree reduction sum, (*sum ...), of the products for the n units (see Fig. 4). There are $1+n(nlog_2 n)$ operations or time order of magnitude $O(n^2)$. This is the same order of magnitude as obtained on a serial machine. Further, the performance of the activation update is a function of the number of interconnects to be processed.

The composite approach forms n weighted sums in nine steps (see Fig. 8): five selection operations; the segmented copy scan before the parallel multiplication; the parallel multiplication; the parallel transfer of the products; and the segmented plus scan, which forms the n sums in one step. This gives the composite activation update a time order of magnitude $O(1)$. Performance is independent of the number of interconnects processed. The next section shows that this is not quite true.

The $n^2$ weights in the model can be updated in three parallel steps using the edge list approach (see Fig. 4). The $n^2$ weights in the model can be updated in eight parallel steps using the composite approach (see Fig. 9). In either case, the weight update operation has a time order of magnitude $O(1)$.

The time complexity results obtained for the composite approach apply to computation of the Lyaponov energy equation (3) and the Hopfield weighting prescription (4), given that pvar structures which can be scanned (see Figs. 1 and 6) are used. The same operations performed serially are time order of magnitude $O(n^2)$.

The above operations all incur a one time overhead cost for generating the addresses in the pointer pvars, used for parallel transfers, and arranging the values in segments for scanning. What the above analysis shows is that time complexity is traded for space complexity. The goal of CM programming is to use as many processors as possible at every step.

## PERFORMANCE COMPARISON

Simulations of a Hopfield spin-glass model[2] were run for six different model sizes over the same number (16,384) of physical CM processors to provide a performance comparison between implementation approaches. The Hopfield network was chosen for the performance comparison because of its simple and well known convergence dynamics and because it uses a small set of pvars which allows a wide range of network sizes (degrees of virtualization) to be run. Twelve treaments are run. Six with the edge list approach and six with the composite approach. Table 3-1 shows the model sizes run for each treatment. Each treatment was run at the virtualization level just necessary to accomodate the number of processors required for each simulation.

Two exemplar patterns are stored. Five test patterns are matched against the two exemplars. Two test patterns have their centers removed, two have a row and column removed, and one is a random pattern. Each exemplar was hand picked and tested to insure that it did not produce cross-talk. The number of rows and columns in the exemplars and patterns increase as the size of the networks for the treatments increases.

Since the performance of the CM is at issue, rather than the performance of the network model used, a simple model and a simple pattern set were chosen to minimize consideration of the influence of model dynamics on performance.

Performance is presented by plotting execution speed versus model size. Size is measured by the number of interconnects in a model. The execution speed metric is interconnects updated per second, $N*(N-1)/t$, where N is the number of units in a model and t is the time used to update the activations for all of the units in a model. All of the units were updated three times for each pattern . Convergence was determined by the output activation remaining stable over the final two updates. The value of t for a treatment is the average of 15 samples of t. Fig. 10 shows the activation update cycle time for both approaches. Fig. 11 shows the interconnect update speed plots for both approaches. The edge list approach is plotted in black. The composite approach is plotted in white. The performance shown excludes overhead for interpretation of the *Lisp instructions. The model size categories for each plot correspond to the model sizes and levels of CM virtualization shown in Table I.

Activation Update Cycle Time vs Model Size

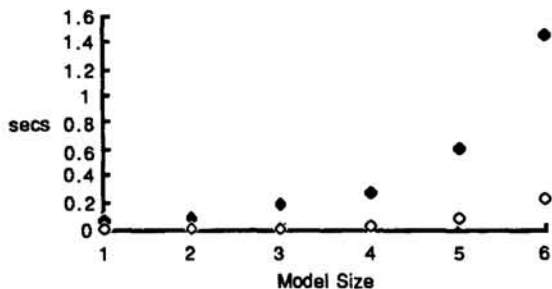

Fig. 10. Activation Update Cycle Times

Interconnect Update Speed Comparison
Edge List Approach vs. Composite Approach

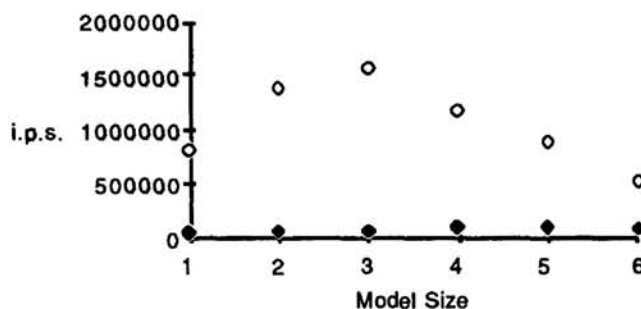

Fig. 11. Edge List Interconnect Update Speeds

Fig. 11 shows an order of magnitude performance difference between the approaches and a roll off in performance for each approach as a function of the number of virtual processors supported by each physical processor. The performance turn around is at 4x virtualization for the edge list approach and 2x virtualization for the composite approach.

## CONCLUSIONS

Representing the interconnect structure of neural network models with mappings defined over the set of fine grain processors provided by the CM architecture provides good performance for a modest programming effort utilizing only a small subset of the instructions provided by *Lisp. Further, the performance will continue to scale up linearly as long as not more than 2x virtualization is required. While the complexity analysis of the composite activation update suggests that its performance should be independent of the number of interconnects to be processed, the performance results show that the performance is indirectly dependent on the number of interconnects to be processed because the level of virtualization required (after the physical processors are exhausted) is dependent on the number of interconnects to be processed and virtualization decreases performance linearly. The complexity analysis of the edge list activation update shows that its performance should be roughly the same as serial implementations on comparable machines. The results suggest that the composite approach is to be prefered over the edge list approach but not be used at a virtualization level higher than 2x.

The mechanism of the composite activation update suggest that hierarchical networks simulated in this fashion will compare in performance to single layer networks because the parallel transfers provide a type of pipeline for activation for synchronously updated hierarchical networks while providing simultaneous activation transfers for asynchronously updated single layer networks. Researchers at Thinking Machines Corporation and the M.I.T. AI Laboratory in Cambridge Mass. use a similar approach for an implementation of NETtalk. Their approach overlaps the weights of connected units and simultaneously pipelines activation forward and error backward.[3]

Performance better than that presented can be gained by translation of the control code from interpreted *Lisp to PARIS and use of the CM2. In addition to not being interpreted, PARIS allows explicit control over important registers that aren't accessable through *Lisp. The CM2 will offer a number of new features which will enhance performance of neural network simulations: a *Lisp compiler, larger processor memory (64K), and floating point processors. The complier and floating point processors will increase execution speeds while the larger processor memories will provide a larger number of virtual processors at the performance turn around points allowing higher performance through higher CM utilization.

## REFERENCES

1. "Introduction to Data Level Parallelism," Thinking Machines Technical Report 86.14, (April 1986).

2. Hopfield, J. J., "Neural networks and physical systems with emergent collective computational abilities," Proc. Natl. Acad. Sci., Vol. 79, (April 1982), pp. 2554-2558.

3. Blelloch, G. and Rosenberg, C. Network Learning on the Connection Machine, M.I.T. Technical Report, 1987.